# On Higher-Order Perceptron Algorithms [*]

**Cristian Brotto**
DICOM, Università dell'Insubria
cristian.brotto@gmail.com

**Claudio Gentile**
DICOM, Università dell'Insubria
claudio.gentile@uninsubria.it

**Fabio Vitale**
DICOM, Università dell'Insubria
fabiovdk@yahoo.com

## Abstract

A new algorithm for on-line learning linear-threshold functions is proposed which efficiently combines second-order statistics about the data with the "logarithmic behavior" of multiplicative/dual-norm algorithms. An initial theoretical analysis is provided suggesting that our algorithm might be viewed as a standard Perceptron algorithm operating on a transformed sequence of examples with improved margin properties. We also report on experiments carried out on datasets from diverse domains, with the goal of comparing to known Perceptron algorithms (first-order, second-order, additive, multiplicative). Our learning procedure seems to generalize quite well, and converges faster than the corresponding multiplicative baseline algorithms.

## 1 Introduction and preliminaries

The problem of on-line learning linear-threshold functions from labeled data is one which have spurred a substantial amount of research in Machine Learning. The relevance of this task from both the theoretical and the practical point of view is widely recognized: On the one hand, linear functions combine flexiblity with analytical and computational tractability, on the other hand, on-line algorithms provide efficient methods for processing massive amounts of data. Moreover, the widespread use of kernel methods in Machine Learning (e.g., [24]) have greatly improved the scope of this learning technology, thereby increasing even further the general attention towards the specific task of incremental learning (generalized) linear functions. Many models/algorithms have been proposed in the literature (stochastic, adversarial, noisy, etc.) : Any list of references would not do justice of the existing work on this subject. In this paper, we are interested in the problem of on-line learning linear-threshold functions from adversarially generated examples. We introduce a new family of algorithms, collectively called the Higher-order Perceptron algorithm (where "higher" means here "higher than one", i.e., "higher than first-order" descent algorithms such as gradient-descent or standard Perceptron-like algorithms"). Contrary to other higher-order algorithms, such as the ridge regression-like algorithms considered in, e.g., [4, 7], Higher-order Perceptron has the ability to put together in a principled and flexible manner second-order statistics about the data with the "logarithmic behavior" of multiplicative/dual-norm algorithms (e.g., [18, 19, 6, 13, 15, 20]). Our algorithm exploits a simplified form of the inverse data matrix, lending itself to be easily combined with the dual norms machinery introduced by [13] (see also [12, 23]). As we will see, this has also computational advantages, allowing us to formulate an efficient (subquadratic) implementation.

Our contribution is twofold. First, we provide an initial theoretical analysis suggesting that our algorithm might be seen as a standard Perceptron algorithm [21] operating on a transformed sequence of examples with improved margin properties. The same analysis also suggests a simple (but principled) way of switching on the fly between higher-order and first-order updates. This is

[*]The authors gratefully acknowledge partial support by the PASCAL Network of Excellence under EC grant n. 506778. This publication only reflects the authors' views.

especially convenient when we deal with kernel functions, a major concern being the sparsity of the computed solution. The second contribution of this paper is an experimental investigation of our algorithm on artificial and real-world datasets from various domains: We compared Higher-order Perceptron to baseline Perceptron algorithms, like the Second-order Perceptron algorithm defined in [7] and the standard ($p$-norm) Perceptron algorithm, as in [13, 12]. We found in our experiments that Higher-order Perceptron generalizes quite well. Among our experimental findings are the following: 1) Higher-order Perceptron is *always* outperforming the corresponding multiplicative ($p$-norm) baseline (thus the stored data matrix is always beneficial in terms of convergence speed); 2) When dealing with Euclidean norms ($p = 2$), the comparison to Second-order Perceptron is less clear and depends on the specific task at hand.

**Learning protocol and notation.** Our algorithm works in the well-known mistake bound model of on-line learning, as introduced in [18, 2], and further investigated by many authors (e.g., [19, 6, 13, 15, 7, 20, 23] and references therein). Prediction proceeds in a sequence of trials. In each trial $t = 1, 2, \ldots$ the prediction algorithm is given an *instance* vector in $\mathbb{R}^n$ (for simplicity, all vectors are normalized, i.e., $||\boldsymbol{x}_t|| = 1$, where $|| \cdot ||$ is the Euclidean norm unless otherwise specified), and then guesses the binary label $y_t \in \{-1, 1\}$ associated with $\boldsymbol{x}_t$. We denote the algorithm's prediction by $\widehat{y}_t \in \{-1, 1\}$. Then the true label $y_t$ is disclosed. In the case when $\widehat{y}_t \neq y_t$ we say that the algorithm has made a prediction *mistake*. We call an *example* a pair $(\boldsymbol{x}_t, y_t)$, and a *sequence of examples* $\mathcal{S}$ any sequence $\mathcal{S} = (\boldsymbol{x}_1, y_1), (\boldsymbol{x}_2, y_2), \ldots, (\boldsymbol{x}_T, y_T)$. In this paper, we are competing against the class of linear-threshold predictors, parametrized by normal vectors $\boldsymbol{u} \in \{\boldsymbol{v} \in \mathbb{R}^n : ||\boldsymbol{v}|| = 1\}$. In this case, a common way of measuring the (relative) prediction performance of an algorithm $A$ is to compare the total number of mistakes of $A$ on $\mathcal{S}$ to some measure of the linear separability of $\mathcal{S}$. One such measure (e.g., [24]) is the cumulative hinge-loss (or soft-margin) $D_\gamma(\boldsymbol{u}; \mathcal{S})$ of $\mathcal{S}$ w.r.t. a linear classifier $\boldsymbol{u}$ at a given margin value $\gamma > 0$: $D_\gamma(\boldsymbol{u}; \mathcal{S}) = \sum_{t=1}^T \max\{0, \gamma - y_t \boldsymbol{u}^\top \boldsymbol{x}_t\}$ (observe that $D_\gamma(\boldsymbol{u}; \mathcal{S})$ vanishes if and only if $\boldsymbol{u}$ separates $\mathcal{S}$ with margin at least $\gamma$.

A *mistake-driven* algorithm $A$ is one which updates its internal state only upon mistakes. One can therefore associate with the run of $A$ on $\mathcal{S}$ a subsequence $\mathcal{M} = \mathcal{M}(\mathcal{S}, A) \subseteq \{1, \ldots, T\}$ of mistaken trials. Now, the standard analysis of these algorithms allows us to restrict the behavior of the comparison class to mistaken trials only and, as a consequence, to refine $D_\gamma(\boldsymbol{u}; \mathcal{S})$ so as to include only trials in $\mathcal{M}$: $D_\gamma(\boldsymbol{u}; \mathcal{S}) = \sum_{t \in \mathcal{M}} \max\{0, \gamma - y_t \boldsymbol{u}^\top \boldsymbol{x}_t\}$. This gives bounds on $A$'s performance relative to the best $\boldsymbol{u}$ over a sequence of examples produced (or, actually, selected) by $A$ during its on-line functioning. Our analysis in Section 3 goes one step further: the number of mistakes of $A$ on $\mathcal{S}$ is contrasted to the cumulative hinge loss of the best $\boldsymbol{u}$ on a transformed sequence $\tilde{\mathcal{S}} = ((\tilde{\boldsymbol{x}}_{i_1}, y_{i_1}), (\tilde{\boldsymbol{x}}_{i_2}, y_{i_2}), \ldots, (\tilde{\boldsymbol{x}}_{i_m}, y_{i_m}))$, where each instance $\boldsymbol{x}_{i_k}$ gets transformed into $\tilde{\boldsymbol{x}}_{i_k}$ through a mapping depending only on the past behavior of the algorithm (i.e., only on examples up to trial $t = i_{k-1}$). As we will see in Section 3, this new sequence $\tilde{\mathcal{S}}$ tends to be "more separable" than the original sequence, in the sense that if $\mathcal{S}$ is linearly separable with some margin, then the transformed sequence $\tilde{\mathcal{S}}$ is likely to be separable with a larger margin.

## 2   The Higher-order Perceptron algorithm

The algorithm (described in Figure 1) takes as input a sequence of nonnegative parameters $\rho_1, \rho_2, \ldots$, and maintains a product matrix $B_k$ (initialized to the identity matrix $I$) and a sum vector $\boldsymbol{v}_k$ (initialized to $\mathbf{0}$). Both of them are indexed by $k$, a counter storing the current number of mistakes (plus one). Upon receiving the $t$-th normalized instance vector $\boldsymbol{x}_t \in \mathbb{R}^n$, the algorithm computes its binary prediction value $\widehat{y}_t$ as the sign of the inner product between vector $B_{k-1} \boldsymbol{v}_{k-1}$ and vector $B_{k-1} \boldsymbol{x}_t$. If $\widehat{y}_t \neq y_t$ then matrix $B_{k-1}$ is updates multiplicatively as $B_k = B_{k-1}(I - \rho_k \boldsymbol{x}_t \boldsymbol{x}_t^\top)$ while vector $\boldsymbol{v}_{k-1}$ is updated additively through the standard Perceptron rule $\boldsymbol{v}_k = \boldsymbol{v}_{k-1} + y_t \boldsymbol{x}_t$. The new matrix $B_k$ and the new vector $\boldsymbol{v}_k$ will be used in the next trial. If $\widehat{y}_t = y_t$ no update is performed (hence the algorithm is mistake driven). Observe that $\rho_k = 0$ for any $k$ makes this algorithm degenerate into the standard Perceptron algorithm [21]. Moreover, one can easily see that, in order to let this algorithm exploit the information collected in the matrix $B$ (and let the algorithm's behavior be substantially different from Perceptron's) we need to ensure $\sum_{k=1}^\infty \rho_k = \infty$. In the sequel, our standard choice will be $\rho_k = c/k$, with $c \in (0, 1)$. See Sections 3 and 4.
Implementing Higher-Order Perceptron can be done in many ways. Below, we quickly describe three of them, each one having its own merits.

1) **Primal version**. We store and update an $n \times n$ matrix $A_k = B_k^\top B_k$ and an $n$-dimensional column

**Parameters:** $\rho_1, \rho_2, ... \in [0, 1)$.
**Initialization:** $B_0 = I$; $\boldsymbol{v}_0 = \boldsymbol{0}$; $k = 1$.
**Repeat for** $t = 1, 2, \ldots, T$ :

1. Get instance $\boldsymbol{x}_t \in \mathbb{R}^n$, $||\boldsymbol{x}_t|| = 1$;
2. Predict $\widehat{y}_t = \text{SGN}(\boldsymbol{w}_{k-1}^\top \boldsymbol{x}_t) \in \{-1, +1\}$, where $\boldsymbol{w}_{k-1} = B_{k-1}^\top B_{k-1} \boldsymbol{v}_{k-1}$;
3. Get label $y_t \in \{-1, +1\}$;
4. if $\widehat{y}_t \neq y_t$ then:
$$\begin{aligned} \boldsymbol{v}_k &= \boldsymbol{v}_{k-1} + y_t\,\boldsymbol{x}_t \\ B_k &= B_{k-1}\,(I - \rho_k\,\boldsymbol{x}_t \boldsymbol{x}_t^\top) \\ k &\leftarrow k + 1. \end{aligned}$$

Figure 1: The Higher-order Perceptron algorithm (for $p = 2$).

vector $\boldsymbol{v}_k$. Matrix $A_k$ is updated as $A_k = A_{k-1} - \rho A_{k-1} \boldsymbol{x} \boldsymbol{x}^\top - \rho \boldsymbol{x} \boldsymbol{x}^\top A_{k-1} + \rho^2 (\boldsymbol{x}^\top A_{k-1} \boldsymbol{x}) \boldsymbol{x} \boldsymbol{x}^\top$, taking $O(n^2)$ operations, while $\boldsymbol{v}_k$ is updated as in Figure 1. Computing the algorithm's margin $\boldsymbol{v}^\top A \boldsymbol{x}$ can then be carried out in time quadratic in the dimension $n$ of the input space.

2) **Dual version**. This implementation allows us the use of kernel functions (e.g., [24]). Let us denote by $X_k$ the $n \times k$ matrix whose columns are the $n$-dimensional instance vectors $\boldsymbol{x}_1, ..., \boldsymbol{x}_k$ where a mistake occurred so far, and $\boldsymbol{y}_k$ be the $k$-dimensional column vector of the corresponding labels. We store and update the $k \times k$ matrix $D_k = [d_{i,j}^{(k)}]_{i,j=1}^k$, the $k \times k$ diagonal matrix $H_k = \text{DIAG}\{\boldsymbol{h}_k\}$, $\boldsymbol{h}_k = (h_1^{(k)}, ..., h_k^{(k)})^\top = X_k^\top X_k \boldsymbol{y}_k$, and the $k$-dimensional column vector $\boldsymbol{g}_k = \boldsymbol{y}_k + D_k H_k \boldsymbol{1}_k$, being $\boldsymbol{1}_k$ a vector of $k$ ones. If we interpret the primal matrix $A_k$ above as $A_k = I + \sum_{i,j=1}^k d_{i,j}^{(k)} \boldsymbol{x}_i \boldsymbol{x}_j^\top$, it is not hard to show that the margin value $\boldsymbol{w}_{k-1}^\top \boldsymbol{x}$ is equal to $\boldsymbol{g}_{k-1}^\top X_{k-1}^\top \boldsymbol{x}$, and can be computed through $O(k)$ extra inner products. Now, on the $k$-th mistake, vector $\boldsymbol{g}$ can be updated with $O(k^2)$ extra inner products by updating $D$ and $H$ in the following way. We let $D_0$ and $H_0$ be empty matrices. Then, given $D_{k-1}$ and $H_{k-1} = \text{DIAG}\{\boldsymbol{h}_{k-1}\}$, we have[1] $D_k = \begin{bmatrix} D_{k-1} & -\rho_k\,\boldsymbol{b}_k \\ -\rho_k\,\boldsymbol{b}_k^\top & d_{k,k}^{(k)} \end{bmatrix}$, where $\boldsymbol{b}_k = D_{k-1} X_{k-1}^\top \boldsymbol{x}_k$, and $d_{k,k}^{(k)} = \rho_k^2\,\boldsymbol{x}_k^\top X_{k-1} \boldsymbol{b}_k - 2\rho_k + \rho_k^2$. On the other hand, $H_k = \text{DIAG}\{\boldsymbol{h}_{k-1} + y_k X_{k-1}^\top \boldsymbol{x}_k \,,\, h_k^{(k)}\}$, with $h_k^{(k)} = \boldsymbol{y}_{k-1}^\top X_{k-1}^\top \boldsymbol{x}_k + y_k$.

Observe that on trials when $\rho_k = 0$ matrix $D_{k-1}$ is padded with a zero row and a zero column. This amounts to say that matrix $A_k = I + \sum_{i,j=1}^k d_{i,j}^{(k)} \boldsymbol{x}_i \boldsymbol{x}_j^\top$, is not updated, i.e., $A_k = A_{k-1}$. A closer look at the above update mechanism allows us to conclude that the overall extra inner products needed to compute $\boldsymbol{g}_k$ is actually quadratic only in the number of past mistaken trials having $\rho_k > 0$. This turns out to be especially important when using a *sparse* version of our algorithm which, on a mistaken trial, decides whether to update both $B$ and $\boldsymbol{v}$ or just $\boldsymbol{v}$ (see Section 4).

3) **Implicit primal version and the dual norms algorithm**. This is based on the simple observation that for any vector $\boldsymbol{z}$ we can compute $B_k \boldsymbol{z}$ by unwrapping $B_k$ as in $B_k \boldsymbol{z} = B_{k-1}(I - \rho \boldsymbol{x} \boldsymbol{x}^\top) \boldsymbol{z} = B_{k-1} \boldsymbol{z}'$, where vector $\boldsymbol{z}' = (\boldsymbol{z} - \rho \boldsymbol{x}\,\boldsymbol{x}^\top \boldsymbol{z})$ can be calculated in time $O(n)$. Thus computing the margin $\boldsymbol{v}^\top B_{k-1}^\top B_{k-1} \boldsymbol{x}$ actually takes $O(nk)$. Maintaining this implicit representation for the product matrix $B$ can be convenient when an efficient dual version is likely to be unavailable, as is the case for the *multiplicative* (or, more generally, dual norms) extension of our algorithm. We recall that a multiplicative algorithm is useful when learning sparse target hyperplanes (e.g., [18, 15, 3, 12, 11, 20]). We obtain a dual norms algorithm by introducing a norm parameter $p \geq 2$, and the associated gradient mapping[2] $\boldsymbol{g} : \boldsymbol{\theta} \in \mathbb{R}^n \to \nabla_{\boldsymbol{\theta}} ||\boldsymbol{\theta}||_p^2 / 2 \in \mathbb{R}^n$. Then, in Figure 1, we normalize instance vectors $\boldsymbol{x}_t$ w.r.t. the $p$-norm, we define $\boldsymbol{w}_{k-1} = B_{k-1}^\top \boldsymbol{g}(B_{k-1} \boldsymbol{v}_{k-1})$, and generalize the matrix update as $B_k = B_{k-1}(I - \rho_k \boldsymbol{x}_t \boldsymbol{g}(\boldsymbol{x}_t)^\top)$. As we will see, the resulting algorithm combines the multiplicative behavior of the $p$-norm algorithms with the "second-order" information contained in the matrix $B_k$. One can easily see that the above-mentioned argument for computing the margin $\boldsymbol{g}(B_{k-1} \boldsymbol{v}_{k-1})^\top B_{k-1} \boldsymbol{x}$ in time $O(nk)$ still holds.

# 3 Analysis

We express the performance of the Higher-order Perceptron algorithm in terms of the hinge-loss behavior of the best linear classifier over the transformed sequence

$$\tilde{S} = (B_0 \boldsymbol{x}_{t(1)}, y_{t(1)}), (B_1 \boldsymbol{x}_{t(2)}, y_{t(2)}), (B_2 \boldsymbol{x}_{t(3)}, y_{t(3)}), \ldots, \tag{1}$$

being $t(k)$ the trial where the $k$-th mistake occurs, and $B_k$ the $k$-th matrix produced by the algorithm. Observe that each feature vector $\boldsymbol{x}_{t(k)}$ gets transformed by a matrix $B_k$ depending on *past* examples only. This is relevant to the argument that $\tilde{S}$ tends to have a larger margin than the original sequence (see the discussion at the end of this section). This neat "on-line structure" does not seem to be shared by other competing higher-order algorithms, such as the "ridge regression-like" algorithms considered, e.g., in [25, 4, 7, 23]. For the sake of simplicity, we state the theorem below only in the case $p = 2$. A more general statement holds when $p \geq 2$.

**Theorem 1** *Let the Higher-order Perceptron algorithm in Figure 1 be run on a sequence of examples* $\mathcal{S} = (\boldsymbol{x}_1, y_1), (\boldsymbol{x}_2, y_2), \ldots, (\boldsymbol{x}_T, y_T)$. *Let the sequence of parameters* $\rho_k$ *satisfy* $0 \leq \rho_k \leq \frac{1-c}{1+|\boldsymbol{v}_{k-1}^\top \boldsymbol{x}_t|}$, *where* $\boldsymbol{x}_t$ *is the $k$-th mistaken instance vector, and* $c \in (0, 1]$. *Then the total number* $m$ *of mistakes satisfies*[3]

$$m \leq \alpha \, \frac{D_\gamma(\boldsymbol{u}; \tilde{S}_c))}{\gamma} + \frac{\alpha^2}{2\gamma^2} + \frac{\alpha}{\gamma} \sqrt{\alpha \, \frac{D_\gamma(\boldsymbol{u}; \tilde{S}_c))}{\gamma} + \frac{\alpha^2}{4\gamma^2}}, \tag{2}$$

*holding for any* $\gamma > 0$ *and any unit norm vector* $\boldsymbol{u} \in \mathbb{R}^n$, *where* $\alpha = \alpha(c) = (2 - c)/c$.

*Proof.* The analysis deliberately mimics the standard Perceptron convergence analysis [21]. We fix an arbitrary sequence $\mathcal{S} = (\boldsymbol{x}_1, y_1), (\boldsymbol{x}_2, y_2), \ldots, (\boldsymbol{x}_T, y_T)$ and let $\mathcal{M} \subseteq \{1, 2, \ldots, T\}$ be the set of trials where the algorithm in Figure 1 made a mistake. Let $t = t(k)$ be the trial where the $k$-th mistake occurred. We study the evolution of $||B_k \boldsymbol{v}_k||^2$ over mistaken trials. Notice that the matrix $B_k^\top B_k$ is positive semidefinite for any $k$. We can write

$$||B_k \boldsymbol{v}_k||^2 = ||B_{k-1} \left(I - \rho_k \, \boldsymbol{x}_t \boldsymbol{x}_t^\top\right) \left(\boldsymbol{v}_{k-1} + y_t \, \boldsymbol{x}_t\right)||^2$$

$$\text{(from the update rule } \boldsymbol{v}_k = \boldsymbol{v}_{k-1} + y_t \, \boldsymbol{x}_t \text{ and } B_k = B_{k-1} \left(I - \rho_k \, \boldsymbol{x}_t \boldsymbol{x}_t^\top\right) \text{)}$$

$$= ||B_{k-1} \boldsymbol{v}_{k-1} + y_t \left(1 - \rho_k y_t \boldsymbol{v}_{k-1} \boldsymbol{x}_t - \rho_k\right) B_{k-1} \boldsymbol{x}_t||^2 \quad \text{(using } ||\boldsymbol{x}_t|| = 1\text{)}$$

$$= ||B_{k-1} \boldsymbol{v}_{k-1}||^2 + 2 \, y_t r_k \, \boldsymbol{v}_{k-1}^\top B_{k-1}^\top B_{k-1} \boldsymbol{x}_t + r_k^2 ||B_{k-1} \boldsymbol{x}_t||^2,$$

where we set for brevity $r_k = 1 - \rho_k y_t \boldsymbol{v}_{k-1} \boldsymbol{x}_t - \rho_k$. We proceed by upper and lower bounding the above chain of equalities. To this end, we need to ensure $r_k \geq 0$. Observe that $y_t \boldsymbol{v}_{k-1} \boldsymbol{x}_t \geq 0$ implies $r_k \geq 0$ if and only if $\rho_k \leq 1/(1 + y_t \boldsymbol{v}_{k-1} \boldsymbol{x}_t)$. On the other hand, if $y_t \boldsymbol{v}_{k-1} \boldsymbol{x}_t < 0$ then, in order for $r_k$ to be nonnegative, it suffices to pick $\rho_k \leq 1$. In both cases $\rho_k \leq (1 - c)/(1 + |\boldsymbol{v}_{k-1} \boldsymbol{x}_t|)$ implies $r_k \geq c > 0$, and also $r_k^2 \leq (1 + \rho_k |\boldsymbol{v}_{k-1} \boldsymbol{x}_t| - \rho_k)^2 \leq (2 - c)^2$. Now, using $y_t \, \boldsymbol{v}_{k-1}^\top B_{k-1}^\top B_{k-1} \boldsymbol{x}_t \leq 0$ (combined with $r_k \geq 0$), we conclude that $||B_k \boldsymbol{v}_k||^2 - ||B_{k-1} \boldsymbol{v}_{k-1}||^2 \leq (2 - c)^2 ||B_{k-1} \boldsymbol{x}_t||^2 = (2 - c)^2 \boldsymbol{x}_t^\top A_{k-1} \boldsymbol{x}_t$, where we set $A_k = B_k^\top B_k$. A simple[4] (and crude) upper bound on the last term follows by observing that $||\boldsymbol{x}_t|| = 1$ implies $\boldsymbol{x}_t^\top A_{k-1} \boldsymbol{x}_t \leq ||A_{k-1}||$, the spectral norm (largest eigenvalue) of $A_{k-1}$. Since a factor matrix of the form $(I - \rho \, \boldsymbol{x} \boldsymbol{x}^\top)$ with $\rho \leq 1$ and $||\boldsymbol{x}|| = 1$ has spectral norm one, we have $\boldsymbol{x}_t^\top A_{k-1} \boldsymbol{x}_t \leq ||A_{k-1}|| \leq \prod_{i=1}^{k-1} ||I - \rho_i \, \boldsymbol{x}_{t(i)} \boldsymbol{x}_{t(i)}^\top||^2 \leq 1$. Therefore, summing over $k = 1, \ldots, m = |\mathcal{M}|$ (or, equivalently, over $t \in \mathcal{M}$) and using $\boldsymbol{v}_0 = \boldsymbol{0}$ yields the upper bound

$$||B_m \boldsymbol{v}_m||^2 \leq (2 - c)^2 \, m. \tag{3}$$

To find a lower bound of the left-hand side of (3), we first pick any unit norm vector $\boldsymbol{u} \in \mathbb{R}^n$, and apply the standard Cauchy-Schwartz inequality: $||B_m \boldsymbol{v}_m|| \geq \boldsymbol{u}^\top B_m \boldsymbol{v}_m$. Then, we observe that for a generic trial $t = t(k)$ the update rule of our algorithm allows us to write

$$\boldsymbol{u}^\top B_k \boldsymbol{v}_k - \boldsymbol{u}^\top B_{k-1} \boldsymbol{v}_{k-1} = r_k \, y_t \, \boldsymbol{u}^\top B_{k-1} \boldsymbol{x}_t \geq r_k \left(\gamma - \max\{0, \gamma - y_t \, \boldsymbol{u}^\top B_{k-1} \boldsymbol{x}_t\}\right),$$

where the last inequality follows from $r_k \geq 0$ and holds for any margin value $\gamma > 0$. We sum

the above over $k = 1, \ldots, m$ and exploit $c \leq r_k \leq 2 - c$ after rearranging terms. This gets $||B_m \boldsymbol{v}_m|| \geq \boldsymbol{u}^\top B_m \boldsymbol{v}_m \geq c\gamma m - (2-c)D_\gamma(\boldsymbol{u}; \tilde{S}_c)$. Combining with (3) and solving for $m$ gives the claimed bound. $\square$

From the above result one can see that our algorithm might be viewed as a standard Perceptron algorithm operating on the transformed sequence $\tilde{S}_c$ in (1). We now give a qualitative argument, which is suggestive of the improved margin properties of $\tilde{S}_c$. Assume for simplicity that all examples $(\boldsymbol{x}_t, y_t)$ in the original sequence are correctly classified by hyperplane $\boldsymbol{u}$ with the same margin $\gamma = y_t \boldsymbol{u}^\top \boldsymbol{x}_t > 0$, where $t = t(k)$. According to Theorem 1, the parameters $\rho_1, \rho_2, \ldots$ should be small positive numbers. Assume, again for simplicity, that all $\rho_k$ are set to the same small enough value $\rho > 0$. Then, up to first order, matrix $B_k = \prod_{i=1}^k (I - \rho \, \boldsymbol{x}_{t(i)} \boldsymbol{x}_{t(i)}^\top)$ can be approximated as $B_k \simeq I - \rho \sum_{i=1}^k \boldsymbol{x}_{t(i)} \boldsymbol{x}_{t(i)}^\top$. Then, to the extent that the above approximation holds, we can write:[5]

$$y_t \, \boldsymbol{u}^\top B_{k-1} \boldsymbol{x}_t = y_t \, \boldsymbol{u}^\top \big(I - \rho \sum_{i=1}^{k-1} \boldsymbol{x}_{t(i)} \boldsymbol{x}_{t(i)}^\top\big) \boldsymbol{x}_t = y_t \, \boldsymbol{u}^\top \big(I - \rho \sum_{i=1}^{k-1} y_{t(i)} \boldsymbol{x}_{t(i)} \, y_{t(i)} \boldsymbol{x}_{t(i)}^\top\big) \boldsymbol{x}_t$$

$$= y_t \, \boldsymbol{u}^\top \boldsymbol{x}_t - \rho \, y_t \big(\sum_{i=1}^{k-1} y_{t(i)} \, \boldsymbol{u}^\top \boldsymbol{x}_{t(i)} \, y_{t(i)} \boldsymbol{x}_{t(i)}^\top\big) \boldsymbol{x}_t = \gamma - \rho \, \gamma \, y_t \, \boldsymbol{v}_{k-1}^\top \boldsymbol{x}_t.$$

Now, $y_t \, \boldsymbol{v}_{k-1}^\top \boldsymbol{x}_t$ is the margin of the (first-order) Perceptron vector $\boldsymbol{v}_{k-1}$ over a mistaken trial for the Higher-order Perceptron vector $\boldsymbol{w}_{k-1}$. Since the two vectors $\boldsymbol{v}_{k-1}$ and $\boldsymbol{w}_{k-1}$ are correlated (recall that $\boldsymbol{v}_{k-1}^\top \boldsymbol{w}_{k-1} = \boldsymbol{v}_{k-1}^\top B_{k-1}^\top B_{k-1} \boldsymbol{v}_{k-1} = ||B_{k-1} \boldsymbol{v}_{k-1}||^2 \geq 0$) the mistaken condition $y_t \, \boldsymbol{w}_{k-1}^\top \boldsymbol{x}_t \leq 0$ is more likely to imply $y_t \, \boldsymbol{v}_{k-1}^\top \boldsymbol{x}_t \leq 0$ than the opposite. This tends to yield a margin larger than the original margin $\gamma$. As we mentioned in Section 2, this is also advantageous from a computational standpoint, since in those cases the matrix update $B_{k-1} \rightarrow B_k$ might be skipped (this is equivalent to setting $\rho_k = 0$), still Theorem 1 would hold.

Though the above might be the starting point of a more thorough theoretical understanding of the margin properties of our algorithm, in this paper we prefer to stop early and leave any further investigation to collecting experimental evidence.

## 4   Experiments

We tested the empirical performance of our algorithm by conducting a number of experiments on a collection of datasets, both artificial and real-world from diverse domains (Optical Character Recognition, text categorization, DNA microarrays). The main goal of these experiments was to compare Higher-order Perceptron (with both $p = 2$ and $p > 2$) to known Perceptron-like algorithms, such as first-order [21] and second-order Perceptron [7], in terms of training accuracy (i.e., convergence speed) and test set accuracy. The results are contained in Tables 1, 2, 3, and in Figure 2.

**Task 1: DNA microarrays and artificial data.** The goal here was to test the convergence properties of our algorithms on sparse target learning tasks. We first tested on a couple of well-known DNA microarray datasets. For each dataset, we first generated a number of random training/test splits (our random splits also included random permutations of the training set). The reported results are averaged over these random splits. The two DNA datasets are: i. The ER+/ER− dataset from [14]. Here the task is to analyze expression profiles of breast cancer and classify breast tumors according to ER (Estrogen Receptor) status. This dataset (which we call the "Breast" dataset) contains 58 expression profiles concerning 3389 genes. We randomly split 1000 times into a training set of size 47 and a test set of size 11. ii. The "Lymphoma" dataset [1]. Here the goal is to separate cancerous and normal tissues in a large B-Cell lymphoma problem. The dataset contains 96 expression profiles concerning 4026 genes. We randomly split the dataset into a training set of size 60 and a test set of size 36. Again, the random split was performed 1000 times. On both datasets, the tested algorithms have been run by cycling 5 times over the current training set. No kernel functions have been used.

We also artificially generated two (moderately) sparse learning problems with margin $\gamma \geq 0.005$ at labeling noise levels $\eta = 0.0$ (linearly separable) and $\eta = 0.1$, respectively. The datasets have been generated at random by first generating two (normalized) target vectors $\boldsymbol{u} \in \{-1, 0, +1\}^{500}$, where the first 50 components are selected independently at random in $\{-1, +1\}$ and the remaining 450

components are 0. Then we set $\eta = 0.0$ for the first target and $\eta = 0.1$ for the second one and, corresponding to each of the two settings, we randomly generated 1000 training examples and 1000 test examples. The instance vectors are chosen at random from $[-1, +1]^{500}$ and then normalized. If $\boldsymbol{u} \cdot \boldsymbol{x}_t \geq \gamma$ then a $+1$ label is associated with $\boldsymbol{x}_t$. If $\boldsymbol{u} \cdot \boldsymbol{x}_t \leq -\gamma$ then a $-1$ label is associated with $\boldsymbol{x}_t$. The labels so obtained are flipped with probability $\eta$. If $|\boldsymbol{u} \cdot \boldsymbol{x}_t| < \gamma$ then $\boldsymbol{x}_t$ is rejected and a new vector $\boldsymbol{x}_t$ is drawn. We call the two datasets "Artificial $_{0.0}$" and "Artificial $_{0.1}$". We tested our algorithms by training over an increasing number of epochs and checking the evolution of the corresponding test set accuracy. Again, no kernel functions have been used.

**Task 2: Text categorization.** The text categorization datasets are derived from the first 20,000 newswire stories in the Reuters Corpus Volume 1 (RCV1, [22]). A standard TF-IDF bag-of-words encoding was used to transform each news story into a normalized vector of real attributes. We built four binary classification problems by "binarizing" consecutive news stories against the four target categories 70, 101, 4, and 59. These are the 2nd, 3rd, 4th, and 5th most frequent[6] categories, respectively, within the first 20,000 news stories of RCV1. We call these datasets RCV1$_x$, where $x = 70, 101, 4, 59$. Each dataset was split into a training set of size 10,000 and a test set of the same size. All algorithms have been trained for a single epoch. We initially tried polynomial kernels, then realized that kernel functions did not significantly alter our conclusions on this task. Thus the reported results refer to algorithms with no kernel functions.

**Task 3: Optical character recognition (OCR).** We used two well-known OCR benchmarks: the USPS dataset and the MNIST dataset [16] and followed standard experimental setups, such as the one in [9], including the *one-versus-rest* scheme for reducing a multiclass problem to a set of binary tasks. We used for each algorithm the standard Gaussian and polynomial kernels, with parameters chosen via 5-fold cross validation on the training set across standard ranges. Again, all algorithms have been trained for a single epoch over the training set. The results in Table 3 only refer to the best parameter settings for each kernel.

**Algorithms.** We implemented the standard Perceptron algorithm (with and without kernels), the Second-order Perceptron algorithm, as described in [7] (with and without kernels), and our Higher-order Perceptron algorithm. The implementation of the latter algorithm (for both $p = 2$ and $p > 2$) was "implicit primal" when tested on the sparse learning tasks, and in dual variables for the other two tasks. When using Second-order Perceptron, we set its parameter $a$ (see [7] for details) by testing on a generous range of values. For brevity, only the settings achieving the best results are reported. On the sparse learning tasks we tried Higher-order Perceptron with norm $p = 2, 4, 7, 10$, while on the other two tasks we set $p = 2$. In any case, for each value of $p$, we set[7] $\rho_k = c/k$, with $c = 0, 0.2, 0.4, 0.6, 0.8$. Since $c = 0$ corresponds to a standard $p$-norm Perceptron algorithm [13, 12] we tried to emphasize the comparison $c = 0$ vs. $c > 0$. Finally, when using kernels on the OCR tasks, we also compared to a *sparse* dual version of Higher-order Perceptron. On a mistaken round $t = t(k)$, this algorithm sets $\rho_k = c/k$ if $y_t \boldsymbol{v}_{k-1} \boldsymbol{x}_t \geq 0$, and $\rho_k = 0$ otherwise (thus, when $y_t \boldsymbol{v}_{k-1} \boldsymbol{x}_t < 0$ the matrix $B_{k-1}$ is not updated). For the sake of brevity, the standard Perceptron algorithm is called FO ("First Order"), the Second-order algorithm is denoted by SO ("Second Order"), while the Higher-order algorithm with norm parameter $p$ and $\rho_k = c/k$ is abbreviated as HO$_p(c)$. Thus, for instance, FO = HO$_2(0)$.

**Results and conclusions.** Our Higher-order Perceptron algorithm seems to deliver interesting results. In *all* our experiments HO$_p(c)$ with $c > 0$ outperforms HO$_p(0)$. On the other hand, the comparison HO$_p(c)$ vs. SO depends on the specific task. On the DNA datasets, HO$_p(c)$ with $c > 0$ is clearly superior in Breast. On Lymphoma, HO$_p(c)$ gets worse as $p$ increases. This is a good indication that, in general, a multiplicative algorithm is not suitable for this dataset. In any case, HO$_2$ turns out to be only slightly worse than SO. On the artificial datasets HO$_p(c)$ with $c > 0$ is *always* better than the corresponding $p$-norm Perceptron algorithm. On the text categorization tasks, HO$_2$ tends to perform better than SO. On USPS, HO$_2$ is superior to the other competitors, while on MNIST it performs similarly when combined with Gaussian kernels (though it turns out to be relatively sparser), while it is slightly inferior to SO when using polynomial kernels. The sparse version of HO$_2$ cuts the matrix updates roughly by half, still maintaining a good performance. In all cases HO$_2$ (either sparse or not) significantly outperforms FO.

In conclusion, the Higher-order Perceptron algorithm is an interesting tool for on-line binary clas-

Table 1: Training and test error on the two datasets "Breast" and "Lymphoma". Training error is the average total number of updates over 5 training epochs, while test error is the average fraction of misclassified patterns in the test set, The results refer to the same training/test splits. For each algorithm, only the best setting is shown (best training and best test setting coincided in these experiments). Thus, for instance, $HO_2$ differs from $FO$ because of the $c$ parameter. We emphasized the comparison $HO_7(0)$ vs. $HO_7(c)$ with best $c$ among the tested values. According to Wilcoxon signed rank test, an error difference of 0.5% or larger might be considered significant. In bold are the smallest figures achieved on each row of the table.

| | | FO | $HO_2$ | $HO_4$ | $HO_7(0)$ | $HO_7$ | $HO_{10}$ | SO |
|---|---|---|---|---|---|---|---|---|
| BREAST | TRAIN | 45.2 | **21.7** | 24.5 | 47.4 | 24.5 | 32.4 | 29.6 |
| | TEST | 23.4% | 16.4% | 13.3% | 15.7% | **12.0%** | 13.5 | 15.0% |
| LYMPHOMA | TRAIN | 22.1 | 19.6 | **18.9** | 23.0 | 20.0 | 23.1 | 19.3 |
| | TEST | 11.8% | 10.0% | 10.0% | 11.5% | 11.5% | 11.9% | **9.6%** |

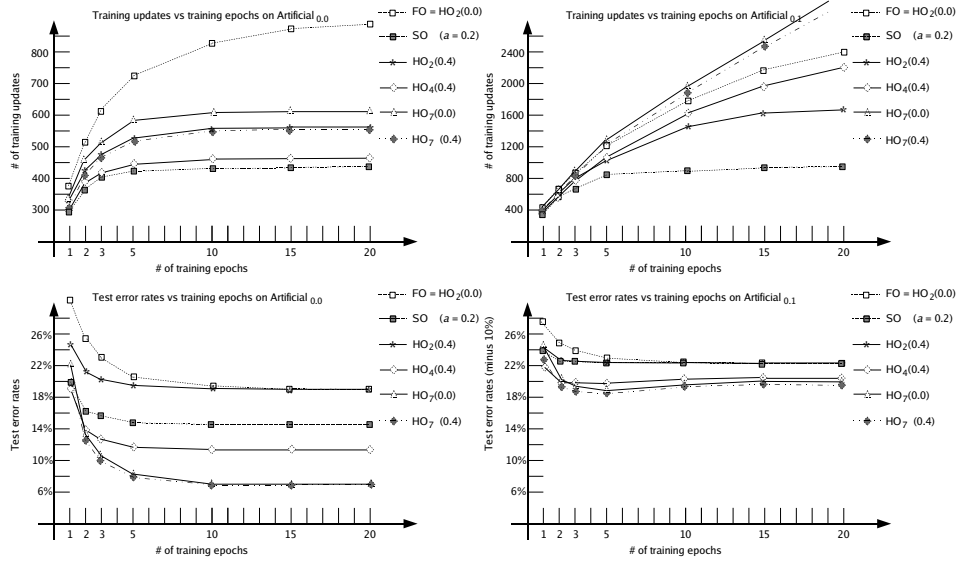

Figure 2: Experiments on the two artificial datasets (Artificial$_{0.0}$, on the left, and Artificial$_{0.1}$, on the right). The plots give training and test behavior as a function of the number of training epochs. Notice that the test set in Artificial$_{0.1}$ is affected by labelling noise of rate 10%. Hence, a visual comparison between the two plots at the bottom can only be made once we shift down the $y$-axis of the noisy plot by 10%. On the other hand, the two training plots (top) are not readily comparable. The reader might have difficulty telling apart the two kinds of algorithms $HO_p(0.0)$ and $HO_p(c)$ with $c > 0$. In practice, the latter turned out to be *always* slightly superior in performance to the former.

sification, having the ability to combine multiplicative (or nonadditive) and second-order behavior into a single inference procedure. Like other algorithms, $HO_p$ can be extended (details omitted due to space limitations) in several ways through known worst-case learning technologies, such as large margin (e.g., [17, 11]), label-efficient/active learning (e.g., [5, 8]), and bounded memory (e.g., [10]).

## Footnotes

[1]Observe that, by construction, $D_k$ is a symmetric matrix.

[2]This mapping has also been used in [12, 11]. Recall that setting $p = O(\log n)$ yields an algorithm similar to Winnow [18]. Also, notice that $p = 2$ yields $\boldsymbol{g} =$ identity.

[3]The subscript $c$ in $\tilde{S}_c$ emphasizes the dependence of the transformed sequence on the choice of $c$. Note that in the special case $c = 1$ we have $\rho_k = 0$ for any $k$ and $\alpha = 1$, thereby recovering the standard Perceptron bound for nonseparable sequences (see, e.g., [12]).

[4]A slightly more refined bound can be derived which depends on the trace of matrices $I - A_k$. Details will be given in the full version of this paper.

[5]Again, a similar argument holds in the more general setting $p \geq 2$. The reader should notice how important the dependence of $B_k$ on the past is to this argument.

[6]We did not use the most frequent category because of its significant overlap with the other ones.

[7]Notice that this setting fulfills the condition on $\rho_k$ stated in Theorem 1.

# References

[1] A. Alizadeh, et al. (2000). Distinct types of diffuse large b-cell lymphoma identified by gene expression profiling. *Nature*, 403, 503–511.

[2] D. Angluin (1988). Queries and concept learning. *Machine Learning*, 2(4), 319–342.

[3] P. Auer & M.K. Warmuth (1998). Tracking the best disjunction. *Machine Learning*, 32(2), 127–150.

[4] K.S. Azoury & M.K. Warmuth (2001). Relative loss bounds for on-line density estimation with the exponential familiy of distributions. *Machine Learning*, 43(3), 211–246.

[5] A. Bordes, S. Ertekin, J. Weston, & L. Bottou (2005). Fast kernel classifiers with on-line and active learning. *JMLR*, 6, 1579–1619.

[6] N. Cesa-Bianchi, Y. Freund, D. Haussler, D.P. Helmbold, R.E. Schapire, & M.K. Warmuth (1997). How to use expert advice. *J. ACM*, 44(3), 427–485.

Table 2: Experimental results on the four binary classification tasks derived from RCV1. "Train" denotes the number of training corrections, while "Test" gives the fraction of misclassified patterns in the test set. Only the results corresponding to the best test set accuracy are shown. In bold are the smallest figures achieved for each of the 8 combinations of dataset ($RCV1_x$, $x = 70, 101, 4, 59$) and phase (training or test).

|  | FO | | HO$_2$ | | SO | |
|---|---|---|---|---|---|---|
|  | TRAIN | TEST | TRAIN | TEST | TRAIN | TEST |
| $RCV1_{70}$ | 993 | 7.20% | 941 | **6.83%** | **880** | 6.95% |
| $RCV1_{101}$ | 673 | 6.39% | **665** | 5.81% | 677 | **5.48%** |
| $RCV1_4$ | 803 | 6.14% | **783** | **5.94%** | 819 | 6.05% |
| $RCV1_{59}$ | 767 | 6.45% | 762 | **6.04%** | **760** | 6.84% |

Table 3: Experimental results on the OCR tasks. "Train" denotes the total number of training corrections, summed over the 10 categories, while "Test" denotes the fraction of misclassified patterns in the test set. Only the results corresponding to the best test set accuracy are shown. For the sparse version of HO$_2$ we also reported (in parentheses) the number of matrix updates during training. In bold are the smallest figures achieved for each of the 8 combinations of dataset (USPS or MNIST), kernel type (Gaussian or Polynomial), and phase (training or test).

|  |  | FO | | HO$_2$ | | Sparse HO$_2$ | | SO | |
|---|---|---|---|---|---|---|---|---|---|
|  |  | TRAIN | TEST | TRAIN | TEST | TRAIN | TEST | TRAIN | TEST |
| USPS | GAUSS | 1385 | 6.53% | **945** | **4.76%** | 965 (440) | 5.13% | 1003 | 5.05% |
|  | POLY | 1609 | 7.37% | 1090 | 5.71% | 1081 (551) | **5.52%** | **1054** | 5.53% |
| MNIST | GAUSS | 5834 | 2.10% | **5351** | **1.79%** | 5363 (2596) | 1.81% | 5684 | 1.82% |
|  | POLY | 8148 | 3.04% | **6404** | 2.27% | 6476 (3311) | 2.28% | 6440 | **2.03%** |

[7] N. Cesa-Bianchi, A. Conconi & C. Gentile (2005). A second-order perceptron algorithm. *SIAM Journal of Computing*, 34(3), 640–668.

[8] N. Cesa-Bianchi, C. Gentile, & L. Zaniboni (2006). Worst-case analysis of selective sampling for linear-threshold algorithms. *JMLR*, 7, 1205–1230.

[9] C. Cortes & V. Vapnik (1995). Support-vector networks. *Machine Learning*, 20(3), 273–297.

[10] O. Dekel, S. Shalev-Shwartz, & Y. Singer (2006). The Forgetron: a kernel-based Perceptron on a fixed budget. *NIPS 18*, MIT Press, pp. 259–266.

[11] C. Gentile (2001). A new approximate maximal margin classification algorithm. *JMLR*, 2, 213–242.

[12] C. Gentile (2003). The Robustness of the $p$-norm Algorithms. *Machine Learning*, 53(3), pp. 265–299.

[13] A.J. Grove, N. Littlestone & D. Schuurmans (2001). General convergence results for linear discriminant updates. *Machine Learning Journal*, 43(3), 173–210.

[14] S. Gruvberger, et al. (2001). Estrogen receptor status in breast cancer is associated with remarkably distinct gene expression patterns. *Cancer Res.*, *61*, 5979–5984.

[15] J. Kivinen, M.K. Warmuth, & P. Auer (1997). The perceptron algorithm vs. winnow: linear vs. logarithmic mistake bounds when few input variables are relevant. *Artificial Intelligence*, 97, 325–343.

[16] Y. Le Cun, et al. (1995). Comparison of learning algorithms for handwritten digit recognition. *ICANN 1995*, pp. 53–60.

[17] Y. Li & P. Long (2002). The relaxed online maximum margin algorithm. *Machine Learning*, 46(1-3), 361–387.

[18] N. Littlestone (1988). Learning quickly when irrelevant attributes abound: a new linear-threshold algorithm. *Machine Learning*, 2(4), 285–318.

[19] N. Littlestone & M.K. Warmuth (1994). The weighted majority algorithm. *Information and Computation*, 108(2), 212–261.

[20] P. Long & X. Wu (2004). Mistake bounds for maximum entropy discrimination. *NIPS 2004*.

[21] A.B.J. Novikov (1962). On convergence proofs on perceptrons. *Proc. of the Symposium on the Mathematical Theory of Automata, vol. XII*, pp. 615–622.

[22] Reuters: 2000. http://about.reuters.com/researchandstandards/corpus/.

[23] S. Shalev-Shwartz & Y. Singer (2006). Online Learning Meets Optimization in the Dual. *COLT 2006*, pp. 423–437.

[24] B. Schoelkopf & A. Smola (2002). *Learning with kernels*. MIT Press.

[25] Vovk, V. (2001). Competitive on-line statistics. International Statistical Review, 69, 213-248.
